# A Neural Network for Real-Time Signal Processing

**Donald B. Malkoff**
General Electric / Advanced Technology Laboratories
Moorestown Corporate Center
Building 145-2, Route 38
Moorestown, NJ 08057

## ABSTRACT

This paper describes a neural network algorithm that (1) performs temporal pattern matching in real-time, (2) is trained on-line, with a single pass, (3) requires only a single template for training of each representative class, (4) is continuously adaptable to changes in background noise, (5) deals with transient signals having low signal-to-noise ratios, (6) works in the presence of non-Gaussian noise, (7) makes use of context dependencies and (8) outputs Bayesian probability estimates. The algorithm has been adapted to the problem of passive sonar signal detection and classification. It runs on a Connection Machine and correctly classifies, within 500 ms of onset, signals embedded in noise and subject to considerable uncertainty.

## 1    INTRODUCTION

This paper describes a neural network algorithm, STOCHASM, that was developed for the purpose of real-time signal detection and classification. Of prime concern was capability for dealing with transient signals having low signal-to-noise ratios (SNR).

The algorithm was first developed in 1986 for real-time fault detection and diagnosis of malfunctions in ship gas turbine propulsion systems (Malkoff, 1987). It subsequently was adapted for passive sonar signal detection and classification. Recently, versions for information fusion and radar classification have been developed.

Characteristics of the algorithm that are of particular merit include the following:

- It performs well in the presence of either Gaussian or non-Gaussian noise, even where the noise characteristics are changing.

- Improved classifications result from temporal pattern matching in real-time, and by taking advantage of input data context dependencies.

- The network is trained on-line. Single exposures of target data require one pass through the network. Target templates, once formed, can be updated on-line.

- Outputs consist of numerical estimates of closeness for each of the template classes, rather than nearest-neighbor "all-or-none" conclusions.

- The algorithm is implemented in parallel code on a Connection Machine.

Simulated signals, embedded in noise and subject to considerable uncertainty, are classified within 500 ms of onset.

## 2   GENERAL OVERVIEW OF THE NETWORK

### 2.1   REPRESENTATION OF THE INPUTS

Sonar signals used for training and testing the neural network consist of pairs of simulated chirp signals that are superimposed and bounded by a Gaussian envelope. The signals are subject to random fluctuations and embedded in white noise. There is considerable overlapping (similarity) of the signal templates. Real data has recently become available for the radar domain.

Once generated, the time series of the sonar signal is subject to special transformations. The outputs of these transformations are the values which are input to the neural network. In addition, several higher-level signal features, for example, zero crossing data, may be simultaneously input to the same network, for purposes of information fusion. The transformations differ from those used in traditional signal processing. They contribute to the real-time performance and temporal pattern matching capabilities of the algorithm by possessing all the following characteristics:

- **Time-Origin Independence:** The sonar input signal is transformed so the resulting time-frequency representation is independent of the starting time of the transient with respect to its position within the observation window (Figure 1). "Observation window" refers to the most recent segment of the sonar time series that is currently under analysis.

- **Translation Independence:** The time-frequency representation obtained by transforming the sonar input transient does not shift from one network input node to another as the transient signal moves across most of the observation window (Figure 1). In other words, not only does the representation remain the same while the transient moves, but its position relative to specific network nodes also does not change. Each given node continues to receive its

usual kind of information about the sonar transient, despite the relative position of the transient in the window. For example, where the transform is an FFT, a specific input layer node will always receive the output of one specific frequency bin, and none other.

Where the SNR is high, translation independence could be accomplished by a simple time-transformation of the representation before sending it to the neural network. This is not possible in conditions where the SNR is sufficiently low that segmentation of the transient becomes impossible using traditional methods such as auto-regressive analysis; it cannot be determined at what time the transient signal originated and where it is in the observation window.

- The representation gains time-origin and translation independence without sacrificing knowledge about the signal's temporal characteristics or its complex infrastructure. This is accomplished by using (1) the absolute value of the Fourier transform (with respect to time) of the spectrogram of the sonar input, or (2) the radar Woodward Ambiguity Function. The derivation and characterization of these methods for representing data is discussed in a separate paper (Malkoff, 1990).

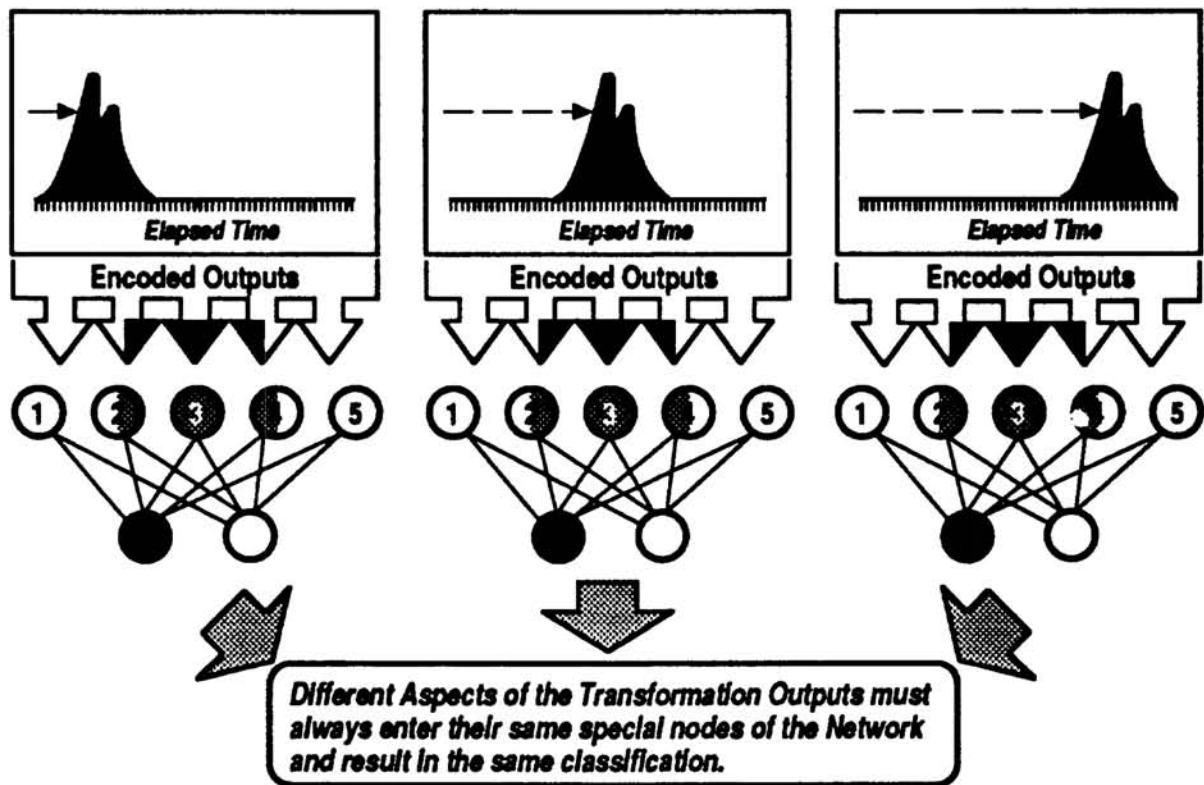

Figure 1: Despite passage of the transient, encoded data enters the same network input nodes (translation independence) and has the same form and output classification (time-origin independence).

## 2.2   THE NETWORK ARCHITECTURE

Sonar data, suitably transformed, enters the network input layer. The input layer serves as a noise filter, or discriminator. The network has two additional layers, the hidden and output layers (Figure 2). Learning of target templates, as well as classification of unknown targets, takes place in a single "feed-forward" pass through these layers. Additional exposures to the same target lead to further enhancement of the template, if training, or refinement of the classification probabilities, if testing.

The hidden layer deals only with data that passes through the input filter. This data predominantly represents a target. Some degree of context dependency evaluation of the data is achieved. Hidden layer data and its permutations are distributed and maintained intact, separate, and transparent. Because of this, credit (error) assignment is easily performed.

In the output layer, evidence is accumulated, heuristically evaluated, and transformed into figures of merit for each possible template class.

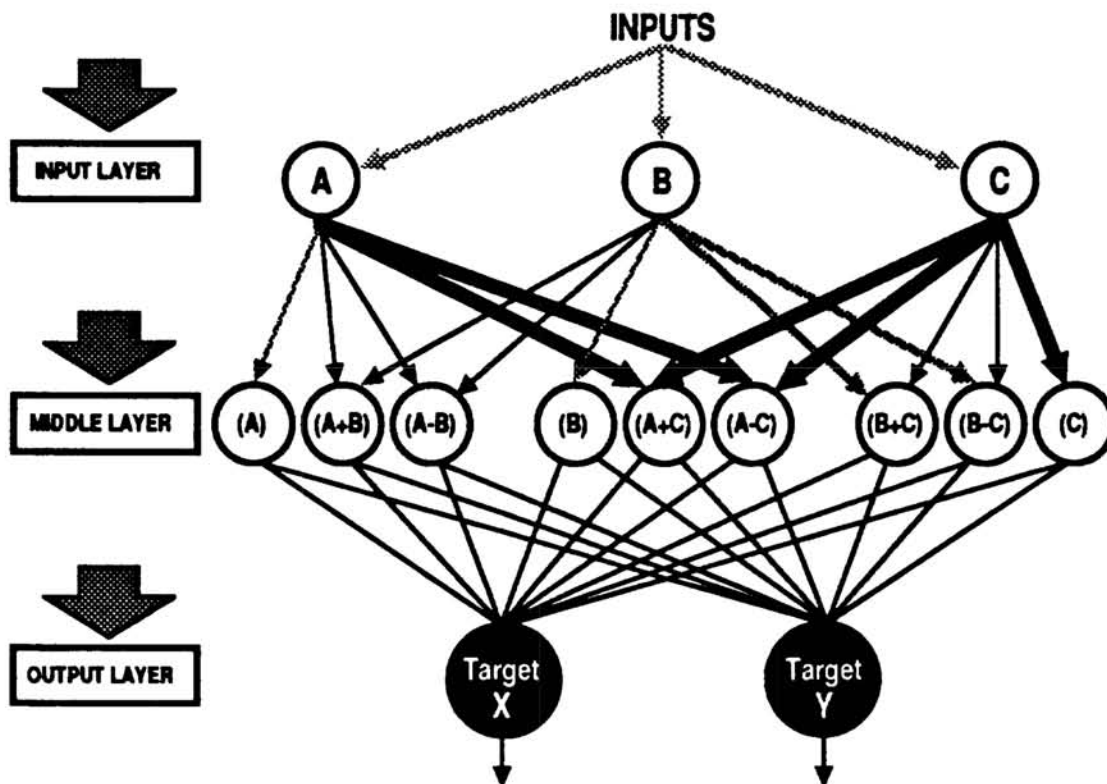

**Figure 2:** STOCHASM network architecture.

### 2.2.1   The Input Layer

Each input layer node receives a succession of samples of a unique part of the sonar representation. This series of samples is stored in a first-in, first-out queue.

With the arrival of each new input sample, the mean and standard deviation of the values in the queue are recomputed at every node. These statistical parameters

are used to detect and extract a signal from the background noise by computing a threshold for each node. Arriving input values that exceed the threshold are passed to the hidden layer and not entered into the queues. Passed values are expressed in terms of $z$-values (the number of standard deviations that the input value differs from the mean of the queued values). Hidden layer nodes receive only data exceeding thresholds; they are otherwise inactive.

### 2.2.2    The Hidden Layer

There are three basic types of hidden layer nodes:

- The first type receive values from only a single input layer node; they reflect absolute changes in an input layer parameter.

- The second type receive values from a pair of inputs where each of those values simultaneously deviates from normal in the same direction.

- The third type receive values from a pair of inputs where each of those values simultaneously deviates from normal in opposite directions.

For $N$ data inputs, there are a total of $N^2$ hidden layer nodes.

Values are passed to the hidden layer only when they exceed the threshold levels determined by the input node queue. The hidden layer values are stored in first-in, first-out queues, like those of the input layer. If the network is in the testing mode, these values represent signals awaiting classification. The mean and standard deviation are computed for each of these queues, and used for subsequent pattern matching. If, instead, the network is in the training mode, the passed values and their statistical descriptors are stored as templates at their corresponding nodes.

### 2.2.3    Pattern Matching Output Layer

Pattern matching consists of computing Bayesian likelihoods for the undiagnosed input relative to each template class. The computation assumes a normal distribution of the values contained within the queue of each hidden layer node. The statistical parameters of the queue representing undiagnosed inputs are matched with those of each of the templates. For example, the number of standard deviations distance between the means of the "undiagnosed" queue and a template queue may be used to demarcate an area under a normal probability distribution. This area is then used as a weight, or measure, for their closeness of match. Note that this computation has a non-linear, sigmoid-shaped output.

The weights for each template are summed across all nodes. Likelihood values are computed for each template. *A priori* data is used where available, and the results normalized for final outputs. The number of computations is minimal and done in parallel; they scale linearly with the number of templates per node. If more computer processing hardware were available, separate processors could be assigned for each template of every node, and computational time would be of constant complexity.

# 3   PERFORMANCE

The sonar version was tested against three sets of totally overlapping double chirp signals, the worst possible case for this algorithm. Where training and testing SNR's differed by a factor of anywhere from 1 to 8, 46 of 48 targets were correctly recognized.

In extensive simulated testing against radar and jet engine modulation data, classifications were better than 95% correct down to -25 dB using the unmodified sonar algorithm.

# 4   DISCUSSION

Distinguishing features of this algorithm include the following capabilities:

- Information fusion.

- Improved classifications.

- Real-time performance.

- Explanation of outputs.

## 4.1   INFORMATION FUSION

In STOCHASM, normalization of the input data facilitates the comparison of separate data items that are diverse in type. This is followed by the fusion, or combination, of all possible pairs of the set of inputs. The resulting combinations are transferred to the hidden layer where they are evaluated and matched with templates. This allows the combining of different features derived either from the same sensor suite or from several different sensor suites. The latter is often one of the most challenging tasks in situation assessment.

## 4.2   IMPROVED CLASSIFICATIONS

### 4.2.1   Multiple Output Weights per Node

In STOCHASM, each hidden layer node receives a single piece of data representing some key feature extracted from the undiagnosed target signal. In contrast, the node has many separate output weights; one for every target template. Each of those output weights represents an actual correlation between the undiagnosed feature data and one of the individual target templates. STOCHASM optimizes the correlations of an unknown input with each possible class. In so doing, it also generates figures of merit (numerical estimates of closeness of match) for ALL the possible target classes, instead of a single "all-or-none" classification.

In more popularized networks, there is only one output weight for each node. Its effectiveness is diluted by having to contribute to the correlation between one undiagnosed feature data and MANY different templates. In order to achieve reasonable classifications, an extra set of input connection weights is employed. The connection

weights provide a somewhat watered-down numerical estimate of the contribution of their particular input data feature to the correct classification, ON THE AVERAGE, of targets representing all possible classes. They employ iterative procedures to compute values for those weights, which prevents real-time training and generates sub-optimal correlations. Moreover, because all of this results in only a single output for each hidden layer node, another set of connection weights between the hidden layer node and each node of the output layer is required to complete the classification process. Since these tend to be fully connected layers, the number of weights and computations is prohibitively large.

### 4.2.2  Avoidance of Nearest-Neighbor Techniques

Some popular networks are sensitive to initial conditions. The determination of the final values of their weights is influenced by the initial values assigned to them. These networks require that, before the onset of training, the values of weights be randomly assigned. Moreover, the classification outcomes of these networks is often altered by changing the order in which training samples are submitted to the network. Networks of this type may be unable to express their conclusions in figures of merit for all possible classes. When inputs to the network share characteristics of more than one target class, these networks tend to gravitate to the classification that initially most closely resembles the input, for an "all-or-none" classification. STOCHASM has none of these drawbacks

### 4.2.3  Noisy Data

The algorithm handles SNR's of lower-than-one and situations where training and testing SNR's differ. Segmentation of one dimensional patterns buried in noise is done automatically. Even the noise itself can be classified. The algorithm can adapt on-line to changing background noise patterns.

### 4.3  REAL-TIME PERFORMANCE

There is no need for back-propagation/gradient-descent methods to set the weights during training. Therefore, no iterations or recursions are required. Only a single feed-forward pass of data through the network is needed for either training or classification. Since the number of nodes, connections, layers, and weights is relatively small, and the algorithm is implemented in parallel, the compute time is fast enough to keep up with real-time in most application domains.

### 4.4  EXPLANATION OF OUTPUTS

There is strict separation of target classification evidence in the nodes of this network. In addition, the evidence is maintained so that positive and negative correlation data is separate and easily accessible. This enables improved credit (error) assignment that leads to more effective classifications and the potential for making available to the operator real-time explanations of program behavior.

## 4.5   FUTURE DIRECTIONS

Previous versions of the algorithm dynamically created, destroyed, or re-arranged nodes and their linkages to optimize the network, minimize computations, and eliminate unnecessary inputs. This algorithm also employed a multi-level hierarchical control system. The control system, on-line and in real-time, adjusted sampling rates and queue lengths, governing when the background noise template is permitted to adapt to current noise inputs, and the rate at which it does so. Future versions of the Connection Machine version will be able to effect the same procedures.

Efforts are now underway to:

1. Improve the temporal pattern matching capabilities.

2. Provide better heuristics for the computation of final figures of merit from the massive amount of positive and negative correlation data resident within the hidden layer nodes.

3. Adapt the algorithm to radar domains where time and spatial warping problems are prominent.

4. Simulate more realistic and complex sonar transients, with the expectation the algorithm will perform better on those targets.

5. Apply the algorithm to information fusion tasks.

### References

Malkoff, D.B., "The Application of Artificial Intelligence to the Handling of Real-Time Sensor Based Fault Detection and Diagnosis," *Proceedings of the Eighth Ship Control Systems Symposium*, Volume 3, Ministry of Defence, The Hague, pp 264-276. Also presented at the Hague, Netherlands, October 8, 1987.

Malkoff, D.B., "A Framework for Real-Time Fault Detection and Diagnosis Using Temporal Data," *The International Journal for Artificial Intelligence in Engineering*, Volume 2, No. 2, pp 97-111, April 1987.

Malkoff, D.B. and L. Cohen, "A Neural Network Approach to the Detection Problem Using Joint Time-Frequency Distributions," *Proceedings of the IEEE 1990 International Conference on Acoustics, Speech, and Signal Processing*, Albuquerque, New Mexico, April 1990 (to appear).
